# Learning to Model Spatial Dependency: Semi-Supervised Discriminative Random Fields

**Chi-Hoon Lee**
Department of Computing Science
University of Alberta
chihoon@cs.ualberta.ca

**Shaojun Wang** *
Department of Computer Science and Engineering
Wright State University
shaojun.wang@wright.edu

**Feng Jiao**
Department of Computing Science
University of Waterloo
fjiao@cs.uwaterloo.ca

**Dale Schuurmans, Russell Greiner**
Department of Computing Science
University of Alberta
{dale, greiner}@cs.ualberta.ca

## Abstract

We present a novel, semi-supervised approach to training discriminative random fields (DRFs) that efficiently exploits labeled and unlabeled training data to achieve improved accuracy in a variety of image processing tasks. We formulate DRF training as a form of MAP estimation that combines conditional loglikelihood on labeled data, given a data-dependent prior, with a conditional entropy regularizer defined on unlabeled data. Although the training objective is no longer concave, we develop an efficient local optimization procedure that produces classifiers that are more accurate than ones based on standard supervised DRF training. We then apply our semi-supervised approach to train DRFs to segment both synthetic and real data sets, and demonstrate significant improvements over supervised DRFs in each case.

## 1 Introduction

Random field models are a popular probabilistic framework for representing complex dependencies in natural image data. The two predominant types of random field models correspond to generative versus discriminative graphical models respectively. Classical Markov random fields (MRFs) [2] follow a traditional generative approach, where one models the *joint* probability of the observed image along with the hidden label field over the pixels. Discriminative random fields (DRFs) [11, 10], on the other hand, directly model the *conditional* probability over the pixel label field given an observed image. In this sense, a DRF is equivalent to a conditional random field [12] defined over a 2-D lattice. Following the basic tenet of Vapnik [18], it is natural to anticipate that learning an accurate joint model should be more challenging than learning an accurate conditional model. Indeed, recent experimental evidence shows that DRFs tend to produce more accurate image labeling models than MRFs, in many applications like gesture recognition [15] and object detection [11, 10, 19, 17].

Although DRFs tend to produce superior pixel labellings to MRFs, partly by relaxing the assumption of conditional independence of observed images given the labels, the approach relies more heavily on supervised training. DRF training typically uses *labeled* image data where each pixel label has been assigned. However, it is considerably more difficult to obtain labeled data for image analysis than for other classification tasks, such as document classification, since hand-labeling the individual pixels of each image is much harder than assigning class labels to objects like text documents.

Recently, semi-supervised training has taken on an important new role in many application areas due to the abundance of unlabeled data. Consequently, many researchers are now working on developing semi-supervised learning techniques for a variety of approaches, including generative models [14], self-learning [5], co-training [3], information-theoretic regularization [6, 8], and graph-based transduction [22, 23, 24]. However, most of these techniques have been developed for univariate classification problems, or class label classification with a structured input [22, 23, 24]. Unfortunately, semi-supervised learning for structured classification problems, where the prediction variables are interdependent in complex ways, have not been as widely studied, with few exceptions [1, 9].

Current work on semi-supervised learning for structured predictors [1, 9] has focused primarily on simple sequence prediction tasks where learning and inference can be efficiently performed using standard dynamic programming. Unfortunately, the problem we address is more challenging, since the spatial correlations in a 2-D grid structure create numerous dependency cycles. That is, our graphical model structure prevents exact inference from being feasible. Kumar et al [10] and Vishwanathan et al [19] argue that learning a model in the context of approximate inference creates a greater risk of the over-fitting and over estimating.

In this paper, we extend the work on semi-supervised learning for sequence predictors [1, 9], particularly the CRF based approach [9], to semi-supervised learning of DRFs. There are several advantages of our approach to semi-supervised DRFs. (1) We inherit the standard advantage of discriminative conditional versus joint model training, while still being able to exploit unlabeled data. (2) The use of unlabeled data enhances our ability to avoid parameter over-fitting and over-estimation in grid based random fields even when using a learner that uses only approximate inference methods. (3) We are still able to model spatial correlations in a 2-D lattice, despite the fact that this introduces dependency cycles in the model. That is, our semi-supervised training procedure can be interpreted as a MAP estimator, where the parameter prior for the model on labeled data is governed by the conditional entropy of the model on unlabeled data. This allows us to learn local potentials that capture spatial correlations while often avoiding local over-estimation. We demonstrate the robustness of our model by applying it to a pixel denoising problem on synthetic images, and also to a challenging real world problem of segmenting tumor in magnetic resonance images. In each case, we have obtained significant improvements over current baselines based on standard DRF training.

## 2   Semi-Supervised DRFs (SSDRFs)

We formulate a new semi-supervised DRF training principle based on the standard supervised formulation of [11, 10]. Let $\mathbf{x}$ be an observed input image, represented by $\mathbf{x} = \{x_i\}_{i \in S}$, where $S$ is a set of the observed image pixels (nodes). Let $\mathbf{y} = \{y_i\}_{i \in S}$ be the joint set of labels over all pixels of an image. For simplicity we assume each component $y_i \in \mathbf{y}$ ranges over binary classes $\mathcal{Y} = \{-1, 1\}$. For example, $\mathbf{x}$ might be a magnetic resonance image of a brain and $\mathbf{y}$ is a realization of a joint labeling over all pixels that indicates whether each pixel is normal or a tumor. In this case, $\mathcal{Y}$ would be the set of pre-defined pixel categories (e.g. tumor versus non-tumor). A DRF is a conditional random field defined on the pixel labels, conditioned on the observation $\mathbf{x}$. More explicitly, the joint distribution over the labels $\mathbf{y}$ given the observations $\mathbf{x}$ is written

$$p_\theta(\mathbf{y}|\mathbf{x}) = \frac{1}{Z_\theta(\mathbf{x})} \exp \Big( \sum_{i \in S} \Phi_\mathbf{w}(y_i, \mathbf{x}) + \sum_{i \in S} \sum_{j \in N_i} \Psi_\boldsymbol{\nu}(y_i, y_j, \mathbf{x}) \Big) \qquad (1)$$

Here $N_i$ denotes the neighboring pixels of $i$. $\Phi_\mathbf{w}(y_i, \mathbf{x}) = \log \Big( \sigma(y_i \mathbf{w}^T \mathbf{h}_i(\mathbf{x}) \Big)$ denotes the node potential at pixel $i$, which quantifies the belief that the class label is $y_i$ for the pre-defined feature vextor $\mathbf{h}_i(\mathbf{x})$, where $\sigma(t) = \frac{1}{1+e^{-t}}$. $\Psi_\boldsymbol{\nu}(y_i, y_j, \mathbf{x}) = y_i y_j \mathbf{v}^T \mu_{ij}(\mathbf{x})$ is an edge potential that captures spatial correlations among neighboring pixels (here, the ones at positions $i$ and $j$), such that $\mu_{ij}(\mathbf{x})$ is the pre-defined feature vector associated with observation $\mathbf{x}$. $Z_\theta(\mathbf{x})$ is the normalizing factor, also known as a (conditional) partition function, which is

$$Z_\theta(\mathbf{x}) = \sum_\mathbf{y} \exp \Big( \sum_{i \in S} \Phi_\mathbf{w}(y_i, \mathbf{x}) + \sum_{i \in S} \sum_{j \in N_i} \Psi_\boldsymbol{\nu}(y_i, y_j, \mathbf{x}) \Big) \qquad (2)$$

Finally, $\theta = (\mathbf{w}, \boldsymbol{\nu})$ are the model parameters. When the edge potentials are set to zero, a DRF yields a standard logistic regression classifier. The potentials in a DRF can use properties of the observed image, and thereby relax the conditional independence assumption of MRFs. Moreover, the edge potentials in a DRF can smooth discontinuities between heterogeneous class pixels, and also correct errors made by the node potentials.

Assume we have a set of independent labeled images, $\mathcal{D}^l = \left( (\mathbf{x}^{(1)}, \mathbf{y}^{(1)})), \cdots, (\mathbf{x}^{(M)}, \mathbf{y}^{(M)}) \right)$, and a set of independent unlabeled images, $\mathcal{D}^u = \left( \mathbf{x}^{(M+1)}, \cdots, \mathbf{x}^{(T)} \right)$. Our goal is to build a DRF model from the combined set of labeled and unlabeled examples, $\mathcal{D}^l \cup \mathcal{D}^u$.

The standard supervised DRF training procedure is based on maximizing the log of the posterior probability of the labeled examples in $\mathcal{D}^l$

$$CL(\theta) \;\; = \;\; \sum_{k=1}^{M} \log P(\mathbf{y}^{(k)}|\mathbf{x}^{(k)}) - \frac{\boldsymbol{\nu}^T \boldsymbol{\nu}}{2\tau^2} \tag{3}$$

A Gaussian prior over the edge parameters $\boldsymbol{\nu}$ is assumed and a uniform prior over parameters $\mathbf{w}$. Here $p(\boldsymbol{\nu}) = \mathcal{N}(\boldsymbol{\nu}; \mathbf{0}, \tau^2 \mathbf{I})$, where $\mathbf{I}$ is the identity matrix. The hyperparameter $\tau^2$ adds a regularization term. In effect, the Gaussian prior introduces a form of regularization to limit over-fitting on rare features and avoid degeneracy in the case of correlated features.

There are a few issues regarding the supervised learning criteria (3). First, the value of $\tau^2$ is critical to the final result, and unfortunately selecting the appropriate $\tau^2$ is a non-trivial task, which in turn makes the learning procedures more challenging and costly [13]. Second, the Gaussian prior is data-independent, and is not associated with either the unlabeled or labeled observations a priori.

Inspired by the work in [8] and [9], we propose a semi-supervised learning algorithm for DRFs that makes full use of the available data by exploiting a form of *entropy regularization* as a prior over the parameters on $D^u$. Specifically, for a semi-supervised DRF, we attempt to find $\theta$ that maximizes the following objective function

$$RL(\theta) \;\; = \;\; \sum_{m=1}^{M} \log p_\theta(\mathbf{y}^{(m)}|\mathbf{x}^{(m)}) + \gamma \sum_{m=M+1}^{T} \sum_{\mathbf{y}} p_\theta(\mathbf{y}|\mathbf{x}^{(\mathbf{m})}) \log \mathbf{p}_\theta(\mathbf{y}|\mathbf{x}^{(\mathbf{m})}) \tag{4}$$

The first term of (4) is the conditional likelihood over the labeled data set $\mathcal{D}^l$, and the second term is a conditional entropy prior over the unlabeled data set $\mathcal{D}^u$, weighted by a tradeoff parameter $\gamma$. The resulting estimate is then formulated as a MAP estimate.

The goal of the objective (4) is to minimize the uncertainty on possible configurations over parameters. That is, minimizing the conditional entropy over unlabeled instances provides more confidence to the algorithm that the hypothetical labellings for the unlabeled data are consistent with the supervised labels, as greater certainty on the estimated labellings coincides with greater conditional likelihood on the supervised labels, and vice versa. This criterion has been shown to be effective for univariate classification [8], and chain structured CRFs [9]; here we apply it to the 2-D lattice case.

## 3  Parameter Estimation

Several factors constrain the form of training algorithm: Because of overhead and the risk of divergence, it was not practical to employ a Newton method. Iterative scaling was not possible because the updates no longer have a closed form. Although the criticism of the gradient descent's principle is well taken, it is the most practical approach we will adopt to optimize the semi-supervised MAP formulation (4) and allows us to improve on standard supervised DRF training.

To formulate a local optimization procedure, we need to compute the gradient of the objective (4) with respect to the parameters. Unfortunately, because of the nonlinear mapping function $\sigma(.)$, we are not able to represent the gradient of objective function as compactly as [9], which was able to express the gradient as a product of the covariance matrix of features and the parameter vector $\theta$. Nevertheless, it is straightforward to show that the derivatives of objective function with respect to the node parameters $\mathbf{w}$ is given by [1]

$$\frac{\partial}{\partial \mathbf{w}} RL(\theta) = \tag{5}$$

$$\sum_{m=1}^{M} \sum_{i \in S^m} \left( y_i^{(m)} \left( 1 - \sigma(y_i^{(m)} \mathbf{w}^T \mathbf{h}_i(\mathbf{x}^{(m)})) \right) - \sum_{\mathbf{y}} p_\theta(\mathbf{y}|\mathbf{x}^{(m)}) y_i \left( 1 - \sigma(y_i \mathbf{w}^T \mathbf{h}_i(\mathbf{x}^{(m)})) \right) \right) \mathbf{h}_i(\mathbf{x}^{(m)})$$

$$+\gamma \sum_{m=M+1}^{T} \sum_{i \in S^m} \left( \sum_{\mathbf{y}} p_\theta(\mathbf{y}|\mathbf{x}^{(m)}) \left( \Phi_{\mathbf{w}}(y_i, \mathbf{x}) + \sum_{j \in N_i} \Psi_{\boldsymbol{\nu}}(y_i, y_j, \mathbf{x}) \right) y_i \left( 1 - \sigma(y_i \mathbf{w}^T \mathbf{h}_i(\mathbf{x}^{(m)})) \right) \right.$$

$$- \left[ \sum_{\mathbf{y}} p_\theta(\mathbf{y}|\mathbf{x}^{(m)}) \left( \Phi_{\mathbf{w}}(y_i, \mathbf{x}) + \sum_{j \in N_i} \Psi_{\boldsymbol{\nu}}(y_i, y_j, \mathbf{x}) \right) \right]$$

$$\left. \left[ \sum_{\mathbf{y}} p_\theta(\mathbf{y}|\mathbf{x}^{(m)}) y_i \left( 1 - \sigma(y_i \mathbf{w}^T \mathbf{h}_i(\mathbf{x}^{(m)})) \right) \right] \right) \mathbf{h}_i(\mathbf{x}^{(m)}),$$

where the first term in (5) is the gradient of the supervised component of the DRF over labeled data, and the second term is the gradient of conditional entropy prior of the DRF over unlabeled data.

Given the lattice structure of the joint labels, it is intractable to compute the exact expectation terms in the above derivatives. It is also intractable to compute the conditional partition function $Z_\theta(\mathbf{x})$. Therefore, as in standard supervised DRFs, we need to incorporate some form of approximation. Following [2, 11, 10], we incorporate the pseudo-likelihood approximation, which assumes that the joint conditional distribution can be approximated as a product of the local posterior probabilities given the neighboring nodes and the observation

$$p_\theta(\mathbf{y}|\mathbf{x}) \quad \approx \quad \prod_{i \in S} p_\theta(y_i|y_{N_i}, \mathbf{x}) \tag{6}$$

$$p_\theta(y_i|y_{N_i}, \mathbf{x}) \quad = \quad \frac{1}{z_i(\mathbf{x})} \exp \left( \Phi_{\mathbf{w}}(y_i, \mathbf{x}) + \sum_{j \in N_i} \Psi_{\boldsymbol{\nu}}(y_i, y_j, \mathbf{x}) \right) \tag{7}$$

Using the factored approximation in (7), we can reformulate the training objective as

$$RL^{PL}(\theta) \quad = \quad \sum_{m=1}^{M} \sum_{i=1}^{S^m} \log p_\theta(\mathbf{y}_i^{(m)}|\mathbf{y}_{N_i}^{(m)}, \mathbf{x}^{(m)}) \tag{8}$$

$$+\gamma \sum_{m=M+1}^{T} \sum_{i=1}^{S^m} \sum_{y_i} p_\theta(y_i|y_{N_i}, \mathbf{x}^{(m)}) \log p_\theta(y_i|y_{N_i} \mathbf{x}^{(m)})$$

Here, the derivative of the second term in (8), with respect to the potential parameters $\mathbf{w}$ and $\boldsymbol{\nu}$, can be reformulated as a factored conditional entropy, yielding

$$\frac{\partial}{\partial \mathbf{w}} RL^{PL}(\theta) \tag{9}$$

$$= \sum_{m=1}^{M} \sum_{i \in S^m} \left( y_i^{(m)} \left( 1 - \sigma(y_i^{(m)} \mathbf{w}^T \mathbf{h}_i(\mathbf{x}^{(m)})) \right) - \sum_{y_i} p_\theta(y_i|y_{N_i}, \mathbf{x}^{(m)}) y_i \left( 1 - \sigma(y_i \mathbf{w}^T \mathbf{h}_i(\mathbf{x}^{(m)})) \right) \right) \mathbf{h}_i(\mathbf{x}^{(m)})$$

$$+\gamma \sum_{m=M+1}^{T} \sum_{i \in S^m} \left( \sum_{y_i} p_\theta(y_i|y_{N_i}, \mathbf{x}^{(m)}) \left( \Phi_{\mathbf{w}}(y_i, \mathbf{x}) + \sum_{j \in N_i} \Psi_{\boldsymbol{\nu}}(y_i, y_j, \mathbf{x}) \right) y_i \left( 1 - \sigma(y_i \mathbf{w}^T \mathbf{h}_i(\mathbf{x}^{(m)})) \right) \right.$$

$$- \left[ \sum_{y_i} p_\theta(y_i|y_{N_i} \mathbf{x}^{(m)}) \left( \Phi_{\mathbf{w}}(y_i, \mathbf{x}) + \sum_{j \in N_i} \Psi_{\boldsymbol{\nu}}(y_i, y_j, \mathbf{x}) \right) \right]$$

$$\left. \left[ \sum_{y_i} p_\theta(y_i|y_{N_i}, \mathbf{x}^{(m)}) y_i \left( 1 - \sigma(y_i \mathbf{w}^T \mathbf{h}_i(\mathbf{x}^{(m)})) \right) \right] \right) \mathbf{h}_i(\mathbf{x}^{(m)})$$

Note that $\frac{\partial}{\partial \boldsymbol{\nu}} RL^{PL}(\theta)$ is computed analogously. Assuming the factorization, the true conditional entropy and feature expectations can be computed in terms of local conditional distributions. This allows us efficiently to approximate the global conditional entropy over unlabeled data. Note that there may be an over-smoothing issue associated with the pseudo-likelihood approximation, as mentioned in [10, 19]. However, due to the fast and stable performance of this approximation in the supervised case [2, 10] we still employ it, but below show that the over-smoothing effect is mitigated by our data-dependent prior in the MAP objective (4).

# 4 Inference

As a result of our formulation, the learning method is tightly coupled with the inference steps. That is, for the unlabeled data, $\mathbf{X}_U$, each time we compute the local conditional covariance (9), we perform inference steps for each node $i$ and its neighboring nodes $N_i$. Our inference is based on iterative conditional modes (ICM) [2], and is given by

$$y_i^* = \underset{y_i \in \mathcal{Y}}{\operatorname{argmax}} \, P(y_i|y_{N_i}, X) \qquad (10)$$

where, for each position $i$, we assume that the labels of all of its neighbors $y' \in N_i$ are fixed. We could alternatively compute the marginal conditional probability $P(y_i|\mathbf{X}) = \sum_{y_{S \setminus i}} P(y_i, y_{S \setminus i}|X)$ for each node using the sum-product algorithm (i.e. loopy belief propagation), which iteratively propagates the belief of each node to its neighbors. Clearly, there are a range of approximation methods available, each entailing different accuracy-complexity tradeoffs. However, we have found that ICM yields good performance at our tasks below, and is probably one of the simplest possible alternatives.

# 5 Experiments

Using standard supervised DRF models, Kumar and Hebert [11, 10] reported interesting experimental results for joint classification tasks on a 2-D lattice, which represents an image with a DRF model. Since labeling image data is expensive and tedious, we believe that better results could be further obtained by formulating a MAP estimation of DRFs by also using the abundant unlabeled image data. In this section, we present a series of experiments on synthetic and real data sets using our novel semi-supervised DRFs(SSDRFs). In order to evaluate our model, we compare the results with those using maximum likelihood estimation of supervised DRFs [11]. There is a major reason that we consider the standard MLE DRF from [11] instead of the parameter regularized DRFs from [10]: that is, we want to show the difference between the ML and MAP principles without using any regularization term that can be problematic [10, 13].

To quantify the performance of each model, we used the Jaccard score $J = \frac{TP}{(TP+FP+FN)}$, where TP denotes true positives, FP false positives, and FN false negatives. Although there are many accuracy measures available, we used this score to penalize the false negatives since many imaging tasks are very imbalanced: that is, only a small percentage of pixels are in the "positive" class. The tradeoff parameter, $\gamma$, was hand-tuned on one held out data set and then held fixed at 0.2 for all of the experiments.

## 5.1 Synthetic image sets

Our primary goal in using synthetic data sets was to demonstrate how well different models classified pixels as a binary classification over a 2-D lattice in the presence of noise. We generated 18 synthetic data sets, each with its own shape. The intensities of pixels in each image were independently corrupted by noise generated from a Gaussian $\mathcal{N}(0, 1)$. Figure 1 shows the results of using supervised DRFs, as well as semi-supervised DRFs. [10, 19] reported over-smoothing effects from the local approximation approach of PL while our experiments indicate that the over-smoothing is caused not only by PL approximation, but also by the sensitivity of the regularization to the parameters. However, using our semi-supervised DRF as a MAP formulation, we have dramatically improved the performance over standard supervised DRF.

Note that the first row in Figure 1 shows good results from the standard DRF, while the oversmoothed outputs are presented in the last row. Although the ML approach may learn proper parameters from some of data sets, unfortunately its performance has not been consistent since the standard DRF's learning of the edge potential tends to be overestimated. For instance, the last row shows that overestimating parameters of the DRF segment almost all pixels into a class due to the complicated edges and structures containing non-target area within the target area, while semi-supervised DRF performance is not degraded at all. Overall, by learning more statistics from unlabeled data, our model dominates the standard DRF in most cases. This is because our MAP formulation avoids the overestimate of potentials and uses the edge potential to correct the errors made by the node potential. Figure 2(a) shows the results over 18 synthetic data sets. Each point above the diagonal line in Figure 2(a) indicates SSDRF producing higher Jaccard scores for a data set. Note that our model stably converged as we increased the ratio $(nU/nL)$ of unlabeled data sets in our learning,

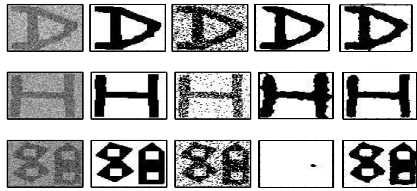

Figure 1: Outputs from synthetic data sets. From left to right: Testing instance, Ground Truth, Logistic Regression (LR), DRF, and SSDRF.

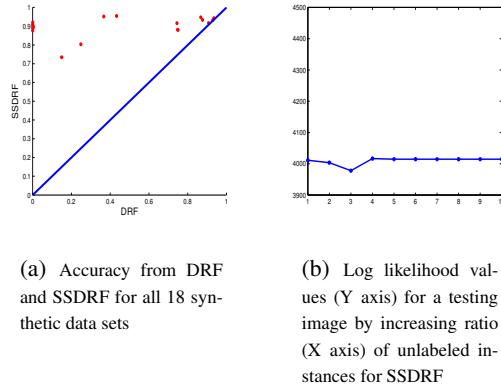

(a) Accuracy from DRF and SSDRF for all 18 synthetic data sets

(b) Log likelihood values (Y axis) for a testing image by increasing ratio (X axis) of unlabeled instances for SSDRF

Figure 2: Accuracy and Convergency

as in Figure 2(b), where $nU$ denotes the number of unlabeled images and $nL$ the number of labeled images. Similar results have also been reported in simple single variable classification task [8].

## 5.2 Brain Tumor Segmentation

We have applied our semi-supervised DRF model to the challenging real world problem of segmenting tumor in medical images. Our goal here is to classify each pixel of an magnetic resonance (MR) image into a pre-defined category: tumor and non-tumor. This is a very important, yet notoriously difficult, task in surgical planning and radiation therapy which currently involves a significant amount of manual work by human medical experts.

We applied three models to the classification of 9 studies from brain tumor MR images. For each study[2], $i$, we divided the MR images into $D_i^L$, $D_i^U$, and $D_i^S$, where an MR image (a.k.a slice) has three modalities available — *T1*, *T2*, and *T1 contrast*. Note that each modality for each slice has $66, 564$ pixels.

As with much of the related work on automatic brain tumor segmentation (such as [7, 21]), our training is based on patient-specific data, where training MR images for a classifier are obtained from the patient to be tested. Note that the training sets and testing sets for a classifier are disjoint. Specifically, LR and DRF takes $D_i^L$ as the training set and $D_i^U$ and $D_i^S$ for testing sets, while SSDRF takes $D_i^L$ and $D_i^U$ for training and $D_i^U$ and $D_i^S$ for testing.

We segmented the "enhancing" tumor area, the region that appears hyper-intense after injecting the contrast agent (we also included non-enhancing areas contained within the enhancing contour). Table 1 and 2 present Jaccard scores of testing $D_i^U$ and $D_i^S$ for each study, $p_i$, respectively. While the standard supervised DRF improves over its degenerate model LR by $1\%$, semi-supervised DRF significantly improves over the supervised DRF by $11\%$, which is significant at $p < 0.00566$ using a paired example *t test*. Considering the fact that MR images contain much noise and the three modalities are not consistent among slices of the same patient, our improvement is considerable. Figure 3 shows the segmentation results by overlaying the testing slices with segmented outputs from the three models. Each row demonstrates the segmentation for a slice, where the white blob areas for the slice correspond to the enhancing tumor area.

## 6 Conclusion

We have proposed a new semi-supervised learning algorithm for DRFs, which was formulated as *MAP* estimation with conditional entropy over unlabeled data as a data-dependent prior regularization. Our approach is motivated by the information-theoretic argument [8, 16] that unlabeled examples can provide the most benefit when classes have small overlap. We introduced a simple approximation approach for this new learning procedure that exploits the local conditional probability to efficiently compute the derivative of objective function.

Table 1: Jaccard Scores for $D_i^U$.

| Studies | Testing from $D_i^U$ | | |
|---|---|---|---|
| | LR | DRF | SSDRF |
| $p_1$ | 53.84 | 59.81 | 59.81 |
| $p_2$ | 83.24 | 83.65 | 84.67 |
| $p_3$ | 30.72 | 30.17 | 75.76 |
| $p_4$ | 72.04 | 76.16 | 79.02 |
| $p_5$ | 73.26 | 73.59 | 75.25 |
| $p_6$ | 88.39 | 89.61 | 87.01 |
| $p_7$ | 69.33 | 69.91 | 75.60 |
| $p_8$ | 58.49 | 58.89 | 73.03 |
| $p_9$ | 60.85 | 56.49 | 83.91 |
| Average | 65.57 | 66.48 | **77.12** |

Table 2: Jaccard Scores for $D_i^S$.

| Studies | Testing from $D_i^S$ | | |
|---|---|---|---|
| | LR | DRF | SSDRF |
| $p_1$ | 68.01 | 68.75 | 68.75 |
| $p_2$ | 69.61 | 69.73 | 70.06 |
| $p_3$ | 23.11 | 21.90 | 71.13 |
| $p_4$ | 56.52 | 63.07 | 68.40 |
| $p_5$ | 51.38 | 52.36 | 51.29 |
| $p_6$ | 85.65 | 86.35 | 85.43 |
| $p_7$ | 66.71 | 68.68 | 70.27 |
| $p_8$ | 44.92 | 45.36 | 73.09 |
| $p_9$ | 21.11 | 20.16 | 38.06 |
| Average | 54.11 | 55.15 | **66.27** |

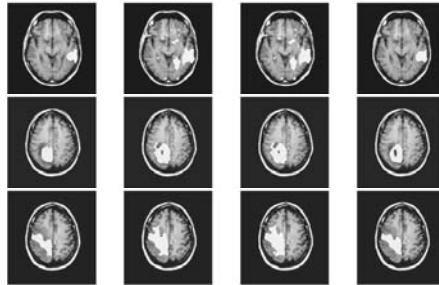

Figure 3: From Left to Right: Human Expert, LR, DRF, and SSDRF

We have applied this new approach to the problem of image pixel classification tasks. By exploiting the availability of auxiliary unlabeled data, we are able to improve the performance of the state of the art supervised DRF approach. Our semi-supervised DRF approach shares all of the benefits of the standard DRF training, including the ability to exploit arbitrary potentials in the presence of dependency cycles, while improving accuracy through the use of the unlabeled data.

The main drawback is the increased training time involved in computing the derivative of the conditional entropy over unlabeled data. Nevertheless, the algorithm is efficient to be trained on unlabeled data sets, and to obtain a significant improvement in classification accuracy over standard supervised training of DRFs as well as iid logistic regression classifiers. To further accelerate the performance with respect to accuracy, we may apply loopy belief propagation [20] or graph-cuts [4] as an inference tool. Since our model is tightly coupled with inference steps during the learning, the proper choice of an inference algorithm will most likely improve segmentation tasks.

**Acknowledgments**

This research is supported by the Alberta Ingenuity Centre for Machine Learning, Cross Cancer Institute, and NSERC. We gratefully acknowledge many helpful suggestions from members of the Brain Tumor Analysis Project, including Dr. A. Murtha and Dr. J Sander.

## Footnotes

*Work done while at University of Alberta

[1]Note that the derivatives of objective function with respect to the edge parameters $\boldsymbol{\nu}$ are computed analogously.

[2]Each study involves a number (typically 21) of images of a single patient – here parallel axial slices through the head.

# References

[1] Y. Altun, D. McAllester, and M. Belkin. Maximum margin semi-supervised learning for structured variables. In *NIPS 18*. 2006.

[2] J. Besag. On the statistical analysis of dirty pictures. *Journal of Royal Statistical Society. Series B*, 48:3:259–302, 1986.

[3] A. Blum and T. Mitchell. Combining labeled and unlabeled data with co-training. In *COLT*, 1998.

[4] Yuri Boykov, Olga Veksler, and Ramin Zabih. Fast approximate energy minimization via graph cuts. In *ICCV (1)*, pages 377–384, 1999.

[5] G. Celeux and G. Govaert. A classification EM algorithm for clustering and two stochastic versions. *Comput. Stat. Data Anal.*, 14(3):315–332, 1992.

[6] A. Corduneanu and T. Jaakkola. Data dependent regularization. In O. Chapelle, B. Schoelkopf, and A. Zien, editors, *Semi-Supervised Learning*. MIT Press, 2006.

[7] C. Garcia and J.A. Moreno. Kernel based method for segmentation and modeling of magnetic resonance images. *LNCS*, 3315:636–645, Oct 2004.

[8] Y. Grandvalet and Y. Bengio. Semi-supervised learning by entropy minimization. In *NIPS 17*, 2004.

[9] F. Jiao, S. Wang, C. Lee, R. Greiner, and D Schuurmans. Semi-supervised conditional random fields for improved sequence segmentation and labeling. In *COLING/ACL*, 2006.

[10] S. Kumar and M. Hebert. Discriminative fields for modeling spatial dependencies in natural images. In *NIPS 16*, 2003.

[11] S. Kumar and M. Hebert. Discriminative random fields: A discriminative framework for contextual interaction in classification. In *CVPR*, 2003.

[12] J. Lafferty, F. Pereira, and A. McCallum. Conditional random fields: Probabilistic models for segmenting and labeling sequence data. In *ICML*, 2001.

[13] C. Lee, R. Greiner, and O. Zaïane. Efficient spatial classification using decoupled conditional random fields. In *10th European Conference on Principles and Practice of Knowledge Discovery in Databases*, pages 272–283, 2006.

[14] K. Nigam, A. McCallum, S. Thrun, and T. Mitchell. Text classification from labeled and unlabeled documents using EM. *Machine Learning*, 39(2/3):103–134, 2000.

[15] A. Quattoni, M. Collins, and T. Darrell. Conditional random fields for object recognition. In *NIPS 17*, 2004.

[16] S. Roberts, R. Everson, and I. Rezek. Maximum certainty data partitioning, 2000.

[17] A. Torralba, K. Murphy, and W. Freeman. Contextual models for object detection using boosted random fields. In *NIPS 17*, 2004.

[18] V. Vapnik. *Statistical Learning Theory*. John-Wiley, 1998.

[19] S.V.N. Vishwanathan, N. Schraudolph, M. Schmidt, and K. Murphy. Accelerated training of conditional random fields with stochastic gradient methods. In *ICML*, 2006.

[20] J. Yedidia, W. Freeman, and Y. Weiss. Generalized belief propagation. In *NIPS 13*, pages 689–695, 2000.

[21] J. Zhang, K. Ma, M.H. Er, and V. Chong. Tumor segmentation from magnetic resonance imaging by learning via one-class support vector machine. *Intl. Workshop on Advanced Image Technology*, 2004.

[22] D. Zhou, O. Bousquet, T. Navin Lal, J. Weston, and B. Schölkopf. Learning with local and global consistency. In *NIPS 16*, 2004.

[23] D. Zhou, J. Huang, and B. Schölkopf. Learning from labeled and unlabeled data on a directed graph. In *ICML*, 2005.

[24] X. Zhu, Z. Ghahramani, and J. Lafferty. Semi-supervised learning using gaussian fields and harmonic functions. In *ICML*, 2003.
